# Multiresolution analysis on the symmetric group

**Risi Kondor and Walter Dempsey**
Department of Statistics and Department of Computer Science
The University of Chicago
`{risi,wdempsey}@uchicago.edu`

## Abstract

There is no generally accepted way to define wavelets on permutations. We address this issue by introducing the notion of coset based multiresolution analysis (CMRA) on the symmetric group, find the corresponding wavelet functions, and describe a fast wavelet transform for sparse signals. We discuss potential applications in ranking, sparse approximation, and multi-object tracking.

## 1 Introduction

A variety of problems in machine learning, from ranking to multi-object tracking, involve inference over permutations. Invariably, the bottleneck in such problems is that the number of permutations grows with $n!$, ruling out the possibility of representing generic functions or distributions over permutations explicitly, as soon as $n$ exceeds about ten or twelve.

Recently, a number of authors have advocated approximations based on a type of generalized Fourier transform [1][2][3][4][5][6]. On the group $\mathbb{S}_n$ of permutations of $n$ objects, this takes the form

$$\widehat{f}(\lambda) = \sum_{\sigma \in \mathbb{S}_n} f(\sigma)\, \rho_\lambda(\sigma), \tag{1}$$

where $\lambda$ plays the role of frequency, while the $\rho_\lambda$ matrix valued functions, called irreducible representations, are similar to the $e^{-i2\pi kx/N}$ factors in ordinary Fourier analysis. It is possible to show that, just as in classical Fourier analysis, the $\widehat{f}(\lambda)$ Fourier matrices correspond to components of $f$ at different levels of smoothness with respect to the underlying permutation topology [2][7]. Ordering the $\lambda$'s from smooth to rough as $\lambda_1 \preccurlyeq \lambda_2 \preccurlyeq \ldots$, one is thus lead to "band-limited" approximations of $f$ via the nested sequence of spaces

$$V_\mu = \{\, f \in \mathbb{R}^{\mathbb{S}_n} \mid \widehat{f}(\lambda) = 0 \ \text{for all} \ \lambda \succ \mu \,\}.$$

While this framework is attractive mathematically, it suffers from the same disease as classical Fourier approximations, namely its inability to handle discontinuities with grace. In applications such as multi-object tracking this is a particularly serious issue, because each observation of the form "object $i$ is at track $j$" introduces a new discontinuity into the assignment distribution, and the resulting Gibbs phenonomenon makes it difficult to ensure even that $f(\sigma)$ remains positive.

The time-honored solution is to use wavelets. However, in the absence of a natural dilation operator, defining wavelets on a discrete space is not trivial. Recently, Gavish et al. defined an analog of Haar wavelets on trees [8], while Coifman and Maggioni [9] and Hammond et al. [10] managed to define wavelets on general graphs. In this paper we attempt to do the same on the much more structured domain of permutations by introducing an altogether new notion of multiresolution analysis, which we call coset-based multiresolution (CMRA).

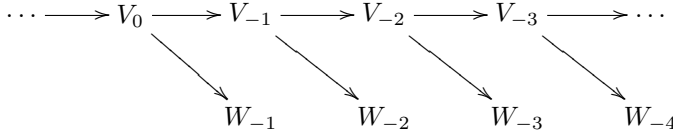

Figure 1: Multiresolution

## 2 Multiresolution analysis and the multiscale structure of $\mathbb{S}_n$

The notion of multiresolution analysis on the real line was first formalized by Mallat [11]: a nested sequence of function spaces

$$\ldots \subset V_{-1} \subset V_0 \subset V_1 \subset V_2 \subset \ldots$$

is said to constitute a multiresolution analysis (MRA) for $L_2(\mathbb{R})$ if it satisfies the following axioms:

MRA1. $\bigcap_k V_k = \{0\}$,

MRA2. $\bigcup_k V_k = L_2(\mathbb{R})$,

MRA3. for any $f \in V_k$ and any $m \in \mathbb{Z}$, the function $f'(x) = f(x - m\, 2^{-k})$ is also in $V_k$,

MRA4. for any $f \in V_k$, the function $f'(x) = f(2x)$, is in $V_{k+1}$.

Setting $V_{k+1} = V_k \oplus W_k$ and starting with, say, $V_\ell$, the process of moving up the chain of spaces can be thought of as splitting $V_\ell$ into a smoother part $V_{\ell-1}$ (called the scaling space) and a rougher part $W_{\ell-1}$ (called the wavelet space), and then repeating this process recursively for $V_{\ell-1}, V_{\ell-2}$, and so on (Figure 1).

To get an actual wavelet transform, one needs to define appropriate bases for the $\{V_i\}$ and $\{W_i\}$ spaces. In the simplest case, a single function $\phi$, called the **scaling function**, is sufficient to generate an orthonormal basis for $V_0$, and a single function $\psi$, called the **mother wavelet** generates an orthonormal basis for $W_0$. In this case, defining $\phi_{k,m}(x) = 2^{k/2}\,\phi(2^k\,x - m)$, and $\psi_{k,m}(x) = 2^{k/2}\,\psi(2^k\,x - m)$, we find that $\{\phi_{k,m}\}_{m \in \mathbb{Z}}$ and $\{\psi_{k,m}\}_{m \in \mathbb{Z}}$ will be orthonormal bases for $V_k$ and $W_k$, respectively. Moreover, $\{\psi_{k,m}\}_{k,m \in \mathbb{Z}}$ is an orthonormal basis for the whole of $L_2(\mathbb{R})$. By the wavelet transform of $f$ we mean its expansion in this basis.

The difficulty in defining multiresolution analysis on discrete spaces is that there is no natural analog of dilation, as required by Mallat's fourth axiom. However, in the specific case of the symmetric group, we do at least have a natural multiscale structure on our domain. Our goal in this paper is to find an analog of Mallat's axioms that can take advantage of this structure.

### 2.1 Two decompositions of $\mathbb{R}^{\mathbb{S}_n}$

A permutation of $n$ objects is a bijective mapping $\{1, 2, \ldots, n\} \rightarrow \{1, 2, \ldots, n\}$. With respect to the natural notion of multiplication $(\sigma_2 \sigma_1)(i) = \sigma_2(\sigma_1(i))$, the $n!$ different permutations of $\{1, \ldots, n\}$ form a **group**, called the **symmetric group** of degree $n$, which we denote $\mathbb{S}_n$.

Our MRA on $\mathbb{S}_n$ is born of the tension between two different ways of carving up $\mathbb{R}^{\mathbb{S}_n}$ into orthogonal sums of subspaces: one corresponding to subdivision in "time", the other in "frequency". The first of these is easier to describe, since it is based on recursively partitioning $\mathbb{S}_n$ according to the hierarchy of sets

$$\begin{aligned} \mathcal{S}_{i_1} &= \{\, \sigma \in \mathbb{S}_n \mid \sigma(n) = i_1 \,\} & i_1 \in \{1, \ldots, n\} \\ \mathcal{S}_{i_1, i_2} &= \{\, \sigma \in \mathbb{S}_n \mid \sigma(n) = i_1,\ \sigma(n-1) = i_2 \,\} & i_1 \neq i_2, \quad i_1, i_2 \in \{1, \ldots, n\}, \end{aligned}$$

and so on, down to sets of the form $\mathcal{S}_{i_1 \ldots i_{n-1}}$, which only have a single element. Intuitively, this tree of nested sets captures the way in which we zoom in on a particular permutation $\sigma$ by first fixing $\sigma(n)$, then $\sigma(n-1)$, etc. (see Figure 2 in Appendix B in the Supplement). From the algebraic point of view, $\mathcal{S}_{i_1, \ldots, i_k}$ is a so-called **(left)** $\mathbb{S}_{n-k}$**–coset**

$$\mu_{i_1, \ldots, i_k} \mathbb{S}_{n-k} := \{\, \mu_{i_1 \ldots i_k} \tau \mid \tau \in \mathbb{S}_{n-k} \,\}, \tag{2}$$

where $\mu_{i_1\ldots i_k}$ is a permutation mapping $n \mapsto i_1, \ldots, n-k+1 \mapsto i_k$. This emphasizes that in some sense each $\mathcal{S}_{i_1,\ldots,i_k}$ is just a "copy" of $\mathbb{S}_{n-k}$ inside $\mathbb{S}_n$. The first important system of subspaces of $\mathbb{R}^{\mathbb{S}_n}$ for our purposes are the **window spaces**

$$S_{i_1\ldots i_k} = \{\, f \mid \mathrm{supp}(f) \subseteq \mathcal{S}_{i_1\ldots i_k} \,\} \qquad 0 \le k \le n-1, \quad \{i_1, \ldots, i_k\} \subseteq \{1, \ldots, n\}\,.$$

Clearly, for any given $k$, $\mathbb{R}^{\mathbb{S}_n} = \bigoplus_{i_1,\ldots,i_k} S_{i_1\ldots i_k}$.

The second system of spaces is related to the behavior of functions under translation. In fact, there are two distinct ways in which a given $f \in \mathbb{R}^{\mathbb{S}_n}$ can be translated by some $\tau \in \mathbb{S}_n$: **left–translation**, $f \mapsto T_\tau f$, where $(T_\tau f)(\sigma) = f(\tau^{-1}\sigma)$, and **right–translation** $f \mapsto T_\tau^R f$, where $(T_\tau^R f)(\sigma) = f(\sigma\tau^{-1})$. For now we focus on the former.

We say that a space $V \subseteq \mathbb{R}^{\mathbb{S}_n}$ is a **left $\mathbb{S}_n$–module** if it is invariant to left-translation in the sense that for any $f \in V$ and $\tau \in \mathbb{S}_n$, $T_\tau f \in V$. A fundamental result in representation theory tells us that if $V$ is **reducible** in the sense that it has a proper subset $V_1$ that is fixed by left-translation, then $V = V_1 \oplus V_2$, where $V_1$ and $V_2$ are *both* (left $\mathbb{S}_n$–)modules. In particular, $\mathbb{R}^{\mathbb{S}_n}$ is a (left $\mathbb{S}_n$–)invariant space, therefore

$$\mathbb{R}^{\mathbb{S}_n} = \bigoplus_{t \in \mathcal{T}_n} M_t \tag{3}$$

for some set $\{M_t\}$ of **irreducible** modules. This is our second important system of spaces.

To understand the interplay between modules and window spaces, observe that each coset $\mu_{i_1\ldots i_k}\mathbb{S}_{n-k}$ has an internal notion of left–translation

$$(T_\tau^{i_1\ldots i_k} f)(\sigma) = f(\mu_{i_1\ldots i_k}\tau^{-1}\mu_{i_1\ldots i_k}^{-1}\sigma), \quad \tau \in \mathbb{S}_{n-k}, \tag{4}$$

which fixes $S_{i_1\ldots i_k}$. Therefore, $S_{i_1\ldots i_k}$ must be decomposable into a sum of irreducible $\mathbb{S}_{n-k}$–modules,

$$S_{i_1\ldots i_k} = \bigoplus_{t \in \mathcal{T}_{n-k}} M_t^{i_1\ldots i_k}. \tag{5}$$

Furthermore, the modules of different window spaces can be defined in such a way that $M_t^{i'_1,\ldots,i'_k} = \mu_{i'_1,\ldots,i'_k}\mu_{i_1\ldots i_k}^{-1} M_t^{i_1\ldots i_k}$. (Note that each $M_t^{i_1\ldots i_k}$ is an $\mathbb{S}_{n-k}$–module in the sense of being invariant to the internal translation action (4), and this action depends on $i_1\ldots i_k$.) Now, for any fixed $t$, the space $U = \bigoplus_{i_1,\ldots,i_k} M_t^{i_1\ldots i_k}$, is fully $\mathbb{S}_n$–invariant, and therefore we must also have $U = \bigoplus_{\alpha \in A} M_\alpha$, where the $M_\alpha$ are now irreducible $\mathbb{S}_n$–modules. Whenever a relationship of this type holds between two sets of irreducible $\mathbb{S}_n$– resp. $\mathbb{S}_{n-k}$–modules, we say that the $\{M_\alpha\}$ modules are **induced** by $\{M_t^{i_1\ldots i_k}\}$.

The situation is complicated by the fact that decompositions like (3) and (5) are not unique. In particular, there is no guarantee that the $\{M_\alpha\}$ induced modules will be amongst the modules featured in (3). However, there is a unique, so-called **adapted** system of modules, for which this issue does not arise. Specifically, if, as is usually done, we let the indexing set $\mathcal{T}_m$ be the set of **Standard Young Tableaux (SYT)** of size $m$ (see Appendix A in the supplementary materials for the exact definition), such as

$$t = \begin{array}{|c|c|c|c|c|} \hline 1 & 3 & 5 & 6 & 7 \\ \hline 2 & 4 \\ \cline{1-2} 8 \\ \cline{1-1} \end{array} \in \mathcal{T}_8,$$

then the adapted modules at different levels of the coset tree are connected via

$$\bigoplus_{i_1\ldots i_k} M_t^{i_1\ldots i_k} = \bigoplus_{t' \in t\uparrow^n} M_{t'} \qquad \forall\, t \in \mathcal{T}_{n-k}, \tag{6}$$

where $t\uparrow^n := \{\, t' \in \mathcal{T}_n \mid t'\downarrow_{n-k} = t \,\}$ and $t'\downarrow_{n-k}$ is the tableau that we get by removing the boxes containing $n-k+1, \ldots, n$ from $t'$. We also extend these relationships to sets in the obvious way: $\mu\downarrow_{n-k} := \{\, t'\downarrow_{n-k} \mid t' \in \mu \,\}$ and $\nu\uparrow^n := \bigcup_{t \in \nu} t\uparrow^n$. We will give an explicit description of the adapted modules in Section 4. For now abstract relationships of the type (6) will suffice.

## 3 Coset based multiresolution analysis on $\mathbb{S}_n$

Our guiding principle in defining an analog of Mallat's axioms for permutations is that the resulting multiresolution analysis should reflect the multiscale structure of the tree of cosets. At the same time, we also want the $\{V_k\}$ spaces to be invariant to translation. Letting $P$ be the projection operator

$$(P_{i_1\ldots i_k}f)(\sigma) := \begin{cases} f(\sigma) & \text{if } \sigma \in \mu_{i_1\ldots i_k}\mathbb{S}_{n-k}, \\ 0 & \text{otherwise,} \end{cases} \tag{7}$$

we propose the following definition.

**Definition 1** *We say that a sequence of spaces $V_0 \subseteq V_1 \subseteq \ldots \subseteq V_{n-1} = \mathbb{R}^{\mathbb{S}_n}$ forms a **left-invariant coset based multiresolution analysis (L-CMRA)** for $\mathbb{S}_n$ if*

*L1. for any $f \in V_k$ and any $\tau \in \mathbb{S}_n$, we have $T_\tau f \in V_k$,*

*L2. if $f \in V_k$, then $P_{i_1\ldots i_{k+1}}f \in V_{k+1}$, for any $i_1, \ldots, i_{k+1}$, and*

*L3. if $g \in V_{k+1}$, then for any $i_1, \ldots, i_{k+1}$ there is an $f \in V_k$ such that $P_{i_1\ldots i_{k+1}}f = g$.*

Given any left-translation invariant space $V_k$, the unique $V_{k+1}$ that satisfies axioms L1–L3 is $V_{k+1} := \bigoplus_{i_1\ldots i_{k+1}} P_{i_1\ldots i_{k+1}} V_k$. Applying this formula recursively, we find that

$$V_k = \bigoplus_{i_1\ldots i_k} P_{i_1\ldots i_k} V_0, \tag{8}$$

so $V_0$ determines the entire sequence of spaces $V_0, V_1, \ldots, V_{n-1}$. In contrast to most classical MRAs, however, this relationship is not bidirectional: $V_k$ does *not* determine $V_0, \ldots, V_{k-1}$.

To gain a better understanding of L-CMRA, we exploit that (by axiom L1) each $V_k$ is $\mathbb{S}_n$–invariant, and is therefore a sum of irreducible $\mathbb{S}_n$–modules. By the following proposition, if $V_0$ is a sum of *adapted* modules, then $V_1, \ldots, V_{n-1}$ are easy to describe.

**Proposition 1** *If $\{M_t\}_{t \in \mathcal{T}_n}$ are the adapted left $\mathbb{S}_n$–modules of $\mathbb{R}^{\mathbb{S}_n}$, and $V_0 = \bigoplus_{t \in \nu_0} M_t$ for some $\nu_0 \subseteq \mathcal{T}_n$, then*

$$V_k = \bigoplus_{t \in \nu_k} M_t, \qquad W_k = \bigoplus_{t \in \nu_{k+1}\setminus\nu_k} M_t, \qquad \text{where} \qquad \nu_k = \nu_0\!\downarrow_{n-k}\!\uparrow^n, \tag{9}$$

*for any $k \in \{0, 1, \ldots, n-1\}$.*

**Proof.** By (6) $P_{i_1\ldots i_k}[\bigoplus_{t' \in t\uparrow^n} M_{t'}] = M_t^{i_1\ldots i_k}$. Therefore, for any $t' \in (t\uparrow^n \cap \nu_0)$ there must be some $f \in M_{t'} \subseteq V_0$ such that for some $i_1 \ldots i_k$, $P_{i_1\ldots i_k}f \in M_t^{i_1\ldots i_k}$ (and $P_{i_1\ldots i_k}f$ is non-zero). By Lemmas 1 and 2 in Appendix D, this implies that $M_t^{i_1\ldots i_k} \subseteq V_k$ for all $i_1 \ldots i_k$. On the other hand, from (6) it is also clear that if $t' \notin \nu_0$, then $M_t^{i_1\ldots i_k} \cap V_k = \{0\}$. Therefore,

$$V_k = \bigoplus_{t \in \nu_0\downarrow_{n-k}} \bigoplus_{i_1\ldots i_k} M_t^{i_1\ldots i_k} = \bigoplus_{t'' \in \nu_0\downarrow_{n-k}\uparrow^n} M_{t''}.$$

The expression for $W_k$ follows from the general formula $V_{k+1} = V_k \oplus W_k$. ∎

**Example 1** The simplest case of L-CMRA is when $\nu_0 = \{\boxed{1\,2\,\cdot\,\cdot\,\cdot\,n}\}$. In this case, setting $m = n - k$, we find that $\nu_0\!\downarrow_m = \{\boxed{1\,2\,\cdot\,\cdot\,\cdot\,m}\}$, and $\nu_k = \nu_0\!\downarrow_m\!\uparrow^n$ is the set of all Young tableaux whose first row starts with the numbers $1, 2, \ldots, m$.

It so happens that $M_{\boxed{1\,2\,\cdot\,\cdot\,m}}^{i_1\ldots i_k}$ is just the trivial invariant subspace of constant functions on $\mu_{i_1\ldots i_k}\mathbb{S}_{n-k}$. Therefore, this instance of L-CMRA is an exact analog of Haar wavelets: $V_k$ will consist of all functions that are constant on each left $\mathbb{S}_{n-k}$–coset. Some more interesting examples of adapted L-CMRAs are described in Appendix C. ⌐

When $V_0$ cannot be written as a direct sum of adapted modules, the analysis becomes significantly more complicated. Due to space limitations, we leave the discussion of this case to the Appendix.

### 3.1 Bi-invariant multiresolution analysis

The left-invariant multiresolution of Definition 1 is appropriate for problems like ranking, where we have a natural permutation invariance with respect to relabeling the objects to be ranked, but not the ranks themselves. In contrast, in problems like multi-object tracking, we want our $V_0 \subset \ldots \subset V_{n-1}$ hierarchy to be invariant on both the left and the right. This leads to the following definition.

**Definition 2** *We say that a sequence of spaces $V_0 \subseteq V_1 \subseteq \ldots \subseteq V_{n-1} = \mathbb{R}^{\mathbb{S}_n}$ forms a **bi-invariant coset based multiresolution analysis (Bi-CMRA)** for $\mathbb{S}_n$ if*

*Bi1. for any $f \in V_k$ and any $\tau \in \mathbb{S}_n$, we have $T_\tau f \in V_k$ and $T_\tau^R f \in V_k$*

*Bi2. if $f \in V_{k-1}$, then $P_{i_1 \ldots i_k} f \in V_k$, for any $i_1, \ldots, i_k$; and*

*Bi3. $V_k$ is the smallest subspace of $\mathbb{R}^{\mathbb{S}_n}$ satisfying Bi1 and Bi2.*

Note that the third axiom had to be modified somewhat compared to Definition 1, but essentially it serves the same purpose as L3.

A subspace $U$ that is invariant to both left- and right-translation (i.e., for any $f \in U$ and any $\sigma, \tau \in \mathbb{S}_n$ both $T_\sigma f \in U$ and $T_\tau^R f \in U$) is called a **two-sided module**. The main reason that Bi-CMRA is easier to describe than L-CMRA is that the irreducible two-sided modules in $\mathbb{R}^{\mathbb{S}_n}$, called **isotypic subspaces**, are *unique*. In particular, the isotypics turn out to be

$$U_\lambda = \bigoplus_{t \in \mathcal{T}_n \,:\, \lambda(t) = \lambda} M_t \qquad \lambda \in \Lambda_n,$$

where $\lambda(t)$ is the vector $(\lambda_1, \ldots, \lambda_p)$ in which $\lambda_i$ is the number of boxes in row $i$ of $t$. For $t$ to be a valid SYT, we must have $\lambda_1 \geq \lambda_2 \geq \ldots \geq \lambda_p \geq 1$, and $\sum_{i=1}^p \lambda_i = n$. We use $\Lambda_n$ to denote the set of all such $p$–tuples, called **integer partitions** of $n$.

Bi-CMRA is a much more constrained framework than L-CMRA because (by axiom Bi1) each $V_k$ space must be of the form $V_k = \bigoplus_{\lambda \in \overline{\nu}_k} U_\lambda$. It should come as no surprise that the way that $\overline{\nu}_0$ determines $\overline{\nu}_1, \ldots, \overline{\nu}_{n-1}$ is related to restriction and extension relationships between partitions. We write $\lambda' \leq \lambda$ if $\lambda'_i \leq \lambda_i$ for all $i$ (assuming $\lambda$ is padded with zeros to make it the same length as $\lambda$), and for $m \leq n$, we define $\lambda \!\downarrow_m := \{ \lambda' \in \Lambda_m \mid \lambda' \leq \lambda \}$, and $\lambda' \!\uparrow^n := \{ \lambda \in \Lambda_n \mid \lambda \geq \lambda' \}$. Again, these operators are extended to sets of partitions by $\mu \!\downarrow_m := \bigcup_{\lambda \in \mu} \lambda \!\downarrow_m$ and $\nu \!\uparrow^n := \bigcup_{\lambda \in \nu} \lambda \!\uparrow^n$. (See Figure 3 in Appendix B.)

**Proposition 2** *Given a set of partitions $\overline{\nu}_0 \subseteq \Lambda_n$, the corresponding Bi-CMRA comprises the spaces*

$$V_k = \bigoplus_{\lambda \in \overline{\nu}_k} U_\lambda, \qquad W_k = \bigoplus_{\lambda \in \overline{\nu}_{k+1} \setminus \overline{\nu}_k} U_\lambda, \qquad where \qquad \overline{\nu}_k = \overline{\nu}_0 \!\downarrow_{n-k}\!\uparrow^n . \qquad (10)$$

*Moreover, any system of spaces satisfying Definition 2 is of this form for some $\overline{\nu}_0 \subseteq \Lambda_n$.*

**Example 2** The simplest case of Bi-CMRA corresponds to taking $\overline{\nu}_0 = \{(n)\}$. In this case $\overline{\nu}_0 \!\downarrow_{n-k} = \{(n-k)\}$, and $\overline{\nu}_k = \{ \lambda \in \Lambda_n \mid \lambda_1 \geq n-k \}$. In Section 6 we discuss that $V_k = \bigoplus_{\lambda \in \nu_k} U_\lambda$ has a clear interpretation as the subspace of $\mathbb{R}^{\mathbb{S}_n}$ determined by up to $k$'th order interactions between elements of the set $\{1, \ldots, n\}$. ⌟

## 4  Wavelets

As mentioned in Section 2, to go from multiresolution analysis to orthogonal wavelets, one needs to define appropriate bases for the spaces $V_0, W_0, W_1, \ldots W_{n-2}$. This can be done via the close connection between irreducible modules and the $\{\rho_\lambda\}$ irreducible representations (irreps), that we encountered in the context of the Fourier transform (1). As explained in Appendix A, each integer partition $\lambda \in \Lambda_n$ has a corresponding irrep $\rho_\lambda \colon \mathbb{S}_n \to \mathbb{R}^{d_\lambda \times d_\lambda}$; the rows and columns of the $\rho_\lambda(\sigma)$ matrices are labeled by the set $\mathcal{T}_\lambda$ of standard Young tableaux of shape $\lambda$; and if the $\rho_\lambda$ are defined according to **Young's Orthogonal Representation (YOR)**, then for any $t \in \mathcal{T}_n$ and $t' \in \mathcal{T}_{\lambda(t)}$, the functions $\varphi_{t'}(\sigma) = [\rho_{\lambda(t)}(\sigma)]_{t',t}$ form a basis for the adapted module $M_t$. Thus, the orthonormal system of functions

$$\phi_{t,t'}(\sigma) = \sqrt{d_\lambda/n!}\, [\rho_\lambda(\sigma)]_{t',t} \qquad t \in \nu_0 \qquad \lambda = \lambda(t) \qquad t' \in \mathcal{T}_\lambda \qquad (11)$$

$$\psi_{t,t'}^k(\sigma) = \sqrt{d_\lambda/n!}\, [\rho_\lambda(\sigma)]_{t',t} \qquad t \in \nu_{k+1} \setminus \nu_k \qquad \lambda = \lambda(t) \quad t' \in \mathcal{T}_\lambda, \qquad (12)$$

seems to be a natural choice of scaling resp. wavelet functions for the L-CMRA of Proposition 1. Similarly, we can take

$$\phi_{t,t'}(\sigma) = \sqrt{d_\lambda/n!}\, [\rho_\lambda(\sigma)]_{t',t} \qquad \lambda \in \overline{\nu}_0 \qquad t, t' \in \mathcal{T}_\lambda \qquad (13)$$

$$\psi_{t,t'}^k(\sigma) = \sqrt{d_\lambda/n!}\, [\rho_\lambda(\sigma)]_{t',t} \qquad \lambda \in \overline{\nu}_{k+1} \setminus \overline{\nu}_k \qquad t, t' \in \mathcal{T}_\lambda, \qquad (14)$$

as a basis for the Bi-CMRA of Proposition 2. Comparing with (1), we find that if we use these bases to compute the wavelet transform of a function, then the wavelet coefficients will just be rescaled versions of specific columns of the Fourier transform. From the computational point of view, this is encouraging, because there are well-known and practical fast Fourier transforms (FFTs) available for $\mathbb{S}_n$ [12][13]. On the other hand, it is also somewhat of a letdown, since it suggests that all that we have gained so far is a way to reinterpret parts of the Fourier transform as wavelet coefficients.

An even more serious concern is that the $\psi_{t,t'}^k$ functions are not at all localized in the spatial domain, largely contradicting the very idea of wavelets. A solution to this dilemma emerges when we consider that since

$$\nu_{k+1} \setminus \nu_k = (\nu_0 \downarrow_{n-k-1} \uparrow^n) \setminus (\nu_0 \downarrow_{n-k} \uparrow^n) = \big((\nu_0 \downarrow_{n-k-1} \uparrow^{n-k}) \setminus (\nu_0 \downarrow_{n-k})\big) \uparrow^n,$$

each of the $W_k$ wavelet spaces of Proposition 1 can be rewritten as

$$W_k = \bigoplus_{i_1 \ldots i_k} \bigoplus_{t \in \omega_k} M_t^{i_1 \ldots i_k} \qquad \omega_k = (\nu_0 \downarrow_{n-k-1} \uparrow^{n-k}) \setminus (\nu_0 \downarrow_{n-k}), \tag{15}$$

and similarly, the wavelet spaces of Proposition 2 can be rewritten as

$$W_k = \bigoplus_{i_1 \ldots i_k} \bigoplus_{\lambda \in \overline{\omega}_k} U_\lambda^{i_1 \ldots i_k} \qquad \overline{\omega}_k = (\overline{\nu}_0 \downarrow_{n-k-1} \uparrow^{n-k}) \setminus (\overline{\nu}_0 \downarrow_{n-k}), \tag{16}$$

where $U_\lambda^{i_1 \ldots i_k}$ are now the "local isotypics" $U_\lambda^{i_1 \ldots i_k} := \bigoplus_{t \in \mathcal{T}_\lambda} M_t^{i_1 \ldots i_k}$. An orthonormal basis for the $M^{i_1 \ldots i_k}$ spaces is provided by the local Fourier basis functions

$$\psi_{t,t'}^{i_1 \ldots i_k}(\sigma) := \begin{cases} \sqrt{d_{\lambda(t)}/(n-k)!} \; [\rho_{\lambda(t)}(\mu_{i_1 \ldots i_k}^{-1} \sigma)]_{t',t} & \sigma \in \mu_{i_1 \ldots i_k} \mathbb{S}_{n-k} \\ 0 & \text{otherwise,} \end{cases} \tag{17}$$

which are localized both in "frequency" and in "space". This basis also affirms the multiscale nature of our wavelet spaces, since projecting onto the wavelet functions $\psi_{t_1,t_1'}^{i_1 \ldots i_k}$ of a specific shape, say, $\lambda_1 = (n-k-2, 2)$ captures very similar information about functions in $S_{i_1 \ldots i_k}$ as projecting onto the analogous $\psi_{t_2,t_2'}^{j_1' \ldots j_{k'}'}$ for functions in $S_{j_1, \ldots, j_{k'}}$ if $t_2$ and $t_2'$ are of shape $\lambda_2 = (n-k'-2, 2)$.

Taking (17) as our wavelet functions, we define the **L-CMRA wavelet transform** of a function $f \colon \mathbb{S}_n \to \mathbb{R}$ as the collection of *column vectors*

$$w_f^*(t) := (\langle f, \phi_{t,t'} \rangle)_{t' \in \lambda(t)}^\top \qquad t \in \nu_0 \tag{18}$$

$$w_f(t; i_1, \ldots, i_k) := (\langle f, \psi_{t,t'}^{i_1 \ldots i_k} \rangle)_{t' \in \lambda(t)}^\top \qquad t \in \omega_k \quad \{i_1, \ldots, i_k\} \subset \{1, \ldots, n\}, \tag{19}$$

where $0 \le k \le n-2$, and $\omega_k$ is as in (15). Similarly, we define the **Bi-CMRA wavelet transform** of $f$ as the collection of *matrices*

$$w_f^*(\lambda) := (\langle f, \phi_{t,t'} \rangle)_{t,t' \in \lambda} \qquad \lambda \in \overline{\nu}_0 \tag{20}$$

$$w_f(\lambda; i_1, \ldots, i_k) := (\langle f, \psi_{t,t'}^{i_1 \ldots i_k} \rangle)_{t,t' \in \lambda} \qquad \lambda \in \overline{\omega}_k \quad \{i_1, \ldots, i_k\} \subset \{1, \ldots, n\}, \tag{21}$$

where $0 \le k \le n-2$, and $\overline{\omega}_k$ is as in (16).

## 4.1 Overcomplete wavelet bases

While the wavelet spaces $W_0, \ldots, W_{k-1}$ of Bi-CMRA are left- and right-invariant, the wavelets (17) still carry the mark of the coset tree, which is not a right-invariant object, since it branches in the specific order $n, n-1, n-2, \ldots$. In contexts where wavelets are used as a means of promoting sparsity, this will bias us towards sparsity patterns that match the particular cosets featured in the coset tree. The only way to avoid this phenomenon is to span $W_0, \ldots, W_{k-1}$ with the overcomplete system of wavelets

$$\psi_{j_1 \ldots j_k, t, t'}^{i_1 \ldots i_k}(\sigma) := \begin{cases} \sqrt{d_{\lambda(t)}/(n-k)!} \; [\rho_{\lambda(t)}(\mu_{i_1 \ldots i_k}^{-1} \sigma \, \mu_{j_1 \ldots j_k})]_{t',t} & \sigma \in \mu_{i_1 \ldots i_k} \mathbb{S}_{n-k} \, \mu_{j_1 \ldots j_k} \\ 0 & \text{otherwise,} \end{cases}$$

where now both $\{i_1, \ldots, i_k\}$ and $\{j_1, \ldots, j_k\}$ are allowed to run over all $k$–element subsets of $\{1, \ldots, n\}$. While sacrificing orthogonality, such a basis is extremely well suited for sparse modeling in various applications.

# 5  Fast wavelet transforms

In the absence of fast wavelet transforms, multiresolution analysis would only be of theoretical interest. Fortunately, our wavelet transforms naturally lend themselves to efficient recursive computation along branches of the coset tree. This is especially attractive when dealing with functions that are sparse, since subtrees that only have zeros at their leaves can be eliminated from the transform altogether.

```
 1: function FastLCWT(f, ν, (i₁ … i_k)) {
 2: if k = n − 1 then
 3:     return(Scaling_ν(v(f)))
 4: end if
 5: v ← 0
 6: for each i_{k+1} ∉ {i₁ … i_k} do
 7:     if P_{i₁…i_{k+1}} f ≠ 0 then
 8:         v ← v + Φ_{i_k}(FastLCWT(f↓_{i₁…i_{k+1}}, ν↓_{n−k−1}, (i₁ … i_{k+1})))
 9:     end if
10: end for
11: output Wavelet_{ν↓_{n−k−1}↑^{n−k}\ν}(v)
12: return Scaling_ν(v) }
```

**Algorithm 1**: A high level description of a recursive algorithm that computes the wavelet transform (18)–(19). The function is called as FastLCWT$(f, \nu_0, ())$. The symbol $\boldsymbol{v}$ stands for the collection of coefficient vectors $\{w_f(t; i_1 \ldots i_k)\}_{t \in \nu\downarrow_{n-k-1}\uparrow^{n-k}}$. The function *Scaling* selects the subset of these vectors that are scaling coefficients, whereas *Wavelet* selects the wavelet coefficients. $f\downarrow_{i_1\ldots i_k} : \mathbb{S}_{n-k} \to \mathbb{R}$ is the restriction of $f$ to $\mu_{i_1\ldots i_k}\mathbb{S}_{n-k}$, i.e., $f\downarrow_{i_1\ldots i_k}(\tau) = f(\mu_{i_1\ldots i_k}\tau)$.

A very high level sketch of the resulting algorithm is given in Algorithm 1, while a more detailed description in terms of actual coefficient matrices is in Appendix E. Bi-CMRA would lead to a similar algorithm, which we omit for brevity. A key component of these algorithms is the function $\Phi_{i_k}$, which serves to convert the coefficient vectors representing any $g \in S_{i_1\ldots i_{k+1}}$ in terms of the basis $\{\psi_{t,t'}^{i_1\ldots i_{k+1}}\}_{t,t'}$ to the coefficient vectors representing the same $g$ in terms of $\{\psi_{t,t'}^{i_1\ldots i_k}\}_{t,t'}$. While in general this can be a complicated and expensive linear transformation, due to the special properties of Young's orthogonal representation, in our case it reduces to

$$w_g(t; i_1 \ldots i_k) = \sqrt{\frac{d_{\lambda'}(n-k)}{d_\lambda}}\, \rho_\lambda([\![i_{k+1}, n-k]\!])\, \left(w_g(t'; i_1 \ldots i_{k+1})\uparrow^t\right), \qquad (22)$$

where $t' = t\downarrow_{n-k-1}$; $\lambda = \lambda(t)$; $\lambda' = \lambda(t')$; $[\![i_{k+1}, k]\!]$ is a special permutation, called a contiguous cycle, that maps $k$ to $i_{k+1}$; and $\uparrow^t$ is a copy operation that promotes its argument to a $d_\lambda$–dimensional vector by

$$\left[w_g(t'; \ldots)\uparrow^t\right]_{t''} = \begin{cases} [w_g(t'; \ldots)]_{t''\downarrow_{n-k-1}} & \text{if } t''\downarrow_{n-k-1} \in \mathcal{T}_{\lambda'} \\ 0 & \text{otherwise.} \end{cases}$$

Clausen's FFT [12] uses essentially the same elementary transformations to compute (1). However, whereas the FFT runs in $O(n^3 n!)$ operations, by working with the local wavelet functions (17) as opposed to (12) and (14), if $f$ is sparse, Algorithm 1 needs only polynomial time.

**Proposition 3** *Given* $f\colon \mathbb{S}_n \to \mathbb{R}$ *such that* $|supp(f)| \le q$, *and* $\nu_0 \subseteq \mathcal{T}_n$, *Algorithm 1 can compute the L-CMRA wavelet coefficients (18)–(19) in* $n^2 N q$ *scalar operations, where* $N = \sum_{t \in \nu_1} d_{\lambda(t)}$. *The analogous Bi-CMRA transform runs in* $n^2 M q$ *time, where* $M = \sum_{\lambda \in \overline{\nu}_1} d_\lambda^2$.

To estimate the $N$ and $M$ constants in this result, note that for partitions with $\lambda_1 \gg \lambda_2, \lambda_3, \ldots$, $d_\lambda = O(n^{n-\lambda_1})$. For example, $d_{(n-1,1)} = n - 1$, $d_{(n-2,2)} = n(n-3)/2$, etc.. The inverse wavelet transforms essentially follow the same computations in reverse and have similar complexity bounds.

# 6 Applications

There is a range of applied problems involving permutations that could benefit from the wavelets defined in this paper. In this section we mention just two potential applications.

## 6.1 Spectral analysis of ranking data

Given a distribution $p$ over permutations, the matrix $\mathcal{M}^k$ of $k$'th order marginals is

$$[\mathcal{M}^k]_{j_1\ldots j_k; i_1\ldots i_k} = p(\,\sigma(i_1)\!=\!j_1,\,\ldots,\,\sigma(i_k)\!=\!j_k\,) = \sum_{\sigma\in\mathcal{S}^{j_1\ldots j_k}_{i_1\ldots i_k}} p(\sigma),$$

where $\mathcal{S}^{j_1\ldots j_k}_{i_1\ldots i_k}$ is the two-sided coset $\mu_{j_1\ldots j_k}\mathbb{S}_{n-k}\mu^{-1}_{i_1\ldots i_k} := \big\{\, \mu_{j_1\ldots j_k}\tau\mu^{-1}_{i_1\ldots i_k} \mid \tau\in\mathbb{S}_{n-k}\,\big\}$. Clearly, these matrices satisfy a number of linear equations, and therefore are redundant. However, it can be shown that for for some appropriate basis transformation matrix $T_k$,

$$\mathcal{M}^k = T_k^\top\Bigg[\bigoplus_{\lambda\in\mathcal{T}_n\,:\,\lambda_1\geq n-k}\widehat{p}(\lambda)\Bigg]T_k,$$

i.e., the Fourier matrices $\{\widehat{p}(\lambda)\}_{\lambda\,:\,\lambda_i=n-k}$ capture exactly the "pure $k$'th order effects" in the distribution $p$. In the spectral analysis of rankings, as advocated, e.g., in [7], there is a lot of emphasis on projecting data to this space, $\mathrm{Marg}_k$, but using an FFT this takes around $O(n^2 n!)$ time. On the other hand, $\mathrm{Marg}_k$ is exactly the wavelet space $W_{k-1}$ of the Bi-CMRA generated by $\overline{\nu}_0 = \{(n)\}$ of Example 2. Therefore, when $p$ is $q$–sparse, noting that $d_{(n-1,1)} = n-1$, by using the methods of the previous section, we can find its projection to each of these spaces in just $O(n^4 q)$ time.

## 6.2 Multi-object tracking

In multi-object tracking, as mentioned in the Introduction, the first few Fourier coefficients $\{\widehat{p}(\lambda)\}_{\lambda\in\xi}$ (w.r.t. the majorizing order on permutations) provide an optimal approximation to the assignment distribution $p$ between targets and tracks in the face of a random noise process [2][1]. However, observing target $i$ at track $j$ will zero out $p$ everywhere outside the coset $\mu_j\mathbb{S}_{n-k}\mu_i^{-1}$, which is difficult for the Fourier approach to handle. In fact, by analogy with (7), denoting the operator that projects to the space of functions supported on this coset by $P_i^j$, the new distribution will just be $P_i^j p$. Thus, if we set $\overline{\nu}_0 = \xi$, after any single observation, our distribution will lie in $V_1$ of the corresponding Bi-CMRA.

Unfortunately, after a second observation, $p$ will fall in $V_2$, etc., leading to a combinatorial explosion in the size of the space needed to represent $p$. However, while each observation makes $p$ less smooth, it also makes it more concentrated, suggesting that this problem is ideally suited to a *sparse* representation in terms of the overcomplete basis functions of Section 4.1. The important departure from the fast wavelet transforms of Section 5 is that now, to find the optimally sparse representation of $p$, we must allow branching to two-sided cosets of the form $\mu_{j_1\ldots j_k}\mathbb{S}_{n-k}\mu_{i_1\ldots i_k}$, which are no longer mutually disjoint.

# 7 Conclusions

Starting from the self-similar structure of the $\mathbb{S}_{n-k}$ coset tree, we developed a framework for wavelet analysis on the symmetric group. Our framework resembles Mallat's multiresolution analysis in its axiomatic foundations, yet is closer to continuous wavelet transforms in its invariance properties. It also has strong ties to the "separation of variables" technique of non-commutative FFTs [14]. In a certain special case we recover the analog of Haar wavelets on the coset tree, In general, wavelets can circumvent the rigidity of the Fourier approach when dealing with functions that are sparse and/or have discontinuities, and, in contrast to the $O(n^2 n!)$ complexity of the best FFTs, for sparse functions and a reasonable choice of $\nu_0$, our fast wavelet transform runs in $O(n^p)$ time for some small $p$. Importantly, wavelets also provide a natural basis for sparse approximations, which have hithero not been explored much in the context of permutations. Finally, much of our framework is applicable not just to the symmetric group, but to other finite groups, as well.

# References

[1] J. Huang, C. Guestrin, and L. Guibas. Fourier Theoretic Probabilistic Inference over Permutations. *Journal of Machine Learning Research*, 10:997–1070, 2009.

[2] R. Kondor, A. Howard, and T. Jebara. Multi-object tracking with representations of the symmetric group. In *Artificial Intelligence and Statistics (AISTATS)*, 2007.

[3] S. Jagabathula and D. Shah. Inferring Rankings under Constrained Sensing. In *In Advances in Neural Information Processing Systems (NIPS)*, 2008.

[4] J. Huang, C. Guestrin, X. Jiang, and L. Guibas. Exploiting Probabilistic Independence for Permutations. In *Artificial Intelligence and Statistics (AISTATS)*, 2009.

[5] X. Jiang, J. Huang, and L. Guibas. Fourier-information duality in the identity management problem. In *In Proceedings of the European Conference on Machine Learning and Principles and Practice of Knowledge Discovery in Databases (ECML PKDD)*, Athens, Greece, September 2011.

[6] D. Rockmore, P. Kostelec, W. Hordijk, and P. F. Stadler. Fast Fourier Transforms for Fitness Landscapes. *Applied and Computational Harmonic Analysis*, 12(1):57–76, 2002.

[7] P. Diaconis. *Group representations in probability and statistics*. Institute of Mathematical Statistics, 1988.

[8] M. Gavish, B. Nadler, and R. R. Coifman. Multiscale Wavelets on Trees, Graphs and High Dimensional Data: Theory and Applications to Semi Supervised Learning. In *International Conference on Machine Learning (ICML)*, 2010.

[9] R. R. Coifman and M. Maggioni. Diffusion wavelets. *Applied and Computational Harmonic Analysis*, 21, 2006.

[10] D. K. Hammond, P. Vandergheynst, and R. Gribonval. Wavelets on graphs via spectral graph theory. *Applied and Computational Harmonic Analysis*, 30:129–150, 2011.

[11] S. G. Mallat. A Theory for Multiresolution Signal Decomposition. *IEEE Transactions on Pattern Analysis and Machine Intelligence*, 11:674–693, 1989.

[12] M. Clausen. Fast generalized Fourier transforms. *Theor. Comput. Sci.*, 67(1):55–63, 1989.

[13] D. Maslen and D. Rockmore. Generalized FFTs – a survey of some recent results. In *Groups and Computation II*, volume 28 of *DIMACS Ser. Discrete Math. Theor. Comput. Sci.*, pages 183–287. AMS, Providence, RI, 1997.

[14] D. K. Maslen and D. N. Rockmore. Separation of Variables and the Computation of Fourier Transforms on Finite Groups, I. *Journal of the American Mathematical Society*, 10:169–214, 1997.

